# Stochastic Online AUC Maximization

**Yiming Ying**[†]**, Longyin Wen**[‡]**, Siwei Lyu**[‡]
[†]Department of Mathematics and Statistics
SUNY at Albany, Albany, NY, 12222, USA
[‡]Department of Computer Science
SUNY at Albany, Albany, NY, 12222, USA

## Abstract

Area under ROC (AUC) is a metric which is widely used for measuring the classification performance for imbalanced data. It is of theoretical and practical interest to develop online learning algorithms that maximizes AUC for large-scale data. A specific challenge in developing online AUC maximization algorithm is that the learning objective function is usually defined over a *pair* of training examples of opposite classes, and existing methods achieves on-line processing with higher space and time complexity. In this work, we propose a new stochastic online algorithm for AUC maximization. In particular, we show that AUC optimization can be equivalently formulated as a convex-concave saddle point problem. From this saddle representation, a stochastic online algorithm (SOLAM) is proposed which has time and space complexity of one datum. We establish theoretical convergence of SOLAM with high probability and demonstrate its effectiveness on standard benchmark datasets.

## 1 Introduction

*Area Under the ROC Curve* (AUC) [8] is a widely used metric for measuring classification performance. Unlike misclassification error that reflects a classifier's ability to classify a single randomly chosen example, AUC concerns the overall performance of a functional family of classifiers and quantifies their ability of correctly ranking any positive instance with regards to a randomly chosen negative instance. Most algorithms optimizing AUC for classification [5, 9, 12, 17] are for batch learning, where we assume all training data are available.

On the other hand, online learning algorithms [1, 2, 3, 16, 19, 22], have been proven to be very efficient to deal with large-scale datasets. However, most studies of online learning focus on the misclassification error or its surrogate loss, in which the objective function depends on a sum of losses over individual examples. It is thus desirable to develop *online* learning algorithms to optimize the AUC metric. The main challenge for an online AUC algorithm is that the objective function of AUC maximization depends on a sum of pairwise losses between instances from different classes which is quadratic in the number of training examples. As such, directly deploying the existing online algorithms will require to store all training data received, making it not feasible for large-scale data analysis.

Several recent works [6, 11, 18, 20, 21] have studied a type of online AUC maximization method that updates the classifier upon the arrival of each new training example. However, this type of algorithms need to access all previous examples at iteration $t$, and has $\mathcal{O}(td)$ space and per-iteration complexity where $d$ is the dimension of the data. The scaling of per-iteration space and time complexity is an undesirable property for online applications that have to use fixed resources. This problem is partially alleviated by the use of buffers of a fixed size $s$ in [11, 21], which reduces the per-iteration space and time complexity to $\mathcal{O}(sd)$. Although this change makes the per-iteration space and time complexity independent of the number of iterations, in practice, to reduce variance in learning performance, the

size of the buffer needs to be set sufficiently large. The work of [6] proposes an alternative method that requires to update and store the first-order (mean) and second-order (covariance) statistics of the training data, and the space and per-iteration complexity becomes $\mathcal{O}(d^2)$. Although this eliminates the needs to access all previous training examples, the per-iteration is now quadratic in data dimension, which makes this method inefficient for high-dimensional data. To this end, the authors of [6] further proposed to approximate the covariance matrices with low-rank random Gaussian matrices. However, the approximation method is not a general solution to the original problem and its convergence was only established under the assumption that the effective numerical rank for the set of covariance matrices is small (*i.e.*, they can be well approximated by low-rank matrices).

In this work, we present a new stochastic online AUC maximization (SOLAM) method associated for the $\ell_2$ loss function. In contrast to existing online AUC maximization methods, *e.g.* [6, 21], SOLAM does not need to store previously received training examples or the covariance matrices, while, at the same time, enjoys a comparable convergence rate, up to a logarithmic term, as in [6, 21]. To our best knowledge, this is the first online learning algorithm for AUC optimization with linear space and per-iteration time complexities of $\mathcal{O}(d)$, which are the same as the online gradient descent algorithm [1, 2, 16, 22] for classification. The key step of SOLAM is to reformulate the original problem as a stochastic saddle point problem [14]. This connection is the foundation of the SOLAM algorithm and its convergence analysis. When evaluating on several standard benchmark datasets, SOLAM achieves performances that are on par with state-of-the-art online AUC optimization methods with significant improvement in running time.

The main contribution of our work can be summarized as follows:

- We provide a new formulation of the AUC optimization problem as stochastic Saddle Point Problem (SPP). This formulation facilitates the development of online algorithms for AUC optimization.

- Our algorithm SOLAM achieves a per-iteration space and time complexity that is linear in data dimensionality.

- Our theoretical analysis provides guarantee of convergence, with high probability, of the proposed algorithm.

## 2 Method

Let the input space $\mathcal{X} \subseteq \mathbb{R}^d$ and the output space $\mathcal{Y} = \{-1, +1\}$. We assume the training data, $\mathbf{z} = \{(x_i, y_i), i = 1, \ldots, n\}$ as *i.i.d.* sample drawn from an unknown distribution $\rho$ on $\mathcal{Z} = \mathcal{X} \times \mathcal{Y}$. The ROC curve is the plot of the true positive rate versus the false positive rate. The area under the ROC curve (AUC) for any scoring function $f : \mathcal{X} \to \mathbb{R}$ is equivalent to the probability of a positive sample ranks higher than a negative sample (e.g. [4, 8]). It is defined as

$$\mathrm{AUC}(f) = \mathrm{Pr}(f(x) \geq f(x')|y = +1, y' = -1), \tag{1}$$

where $(x, y)$ and $(x', y')$ are independent drawn from $\rho$. The target of AUC maximization is to find the optimal decision function $f$:

$$\arg \max_f \mathrm{AUC}(f) = \arg \min_f \mathrm{Pr}(f(x) < f(x')|y = 1, y' = -1)$$

$$= \arg \min_f \mathbb{E}\Big[\mathbb{I}_{[f(x') - f(x) > 0]}\big|y = 1, y' = -1\Big], \tag{2}$$

where $\mathbb{I}(\cdot)$ is the indicator function that takes value 1 if the argument is true and 0 otherwise. Let $p = \mathrm{Pr}(y = 1)$. For any random variable $\xi(z)$, recall that its conditional expectation is defined by $\mathbb{E}[\xi(z)|y = 1] = \frac{1}{p} \iint \xi(z)\mathbb{I}_{y=1} d\rho(z)$. Since $\mathbb{I}(\cdot)$ is not continuous, it is often replaced by its convex surrogates. Two common choices are the $\ell_2$ loss $(1 - (f(x) - f(x')))^2$ or the hinge loss $\big(1 - (f(x) - f(x'))\big)_+$. In this work, we use the $\ell_2$, as it has been shown to be statistically consistent with AUC while the hinge loss is not [6, 7]. We also restrict our interests to the family of linear functions, *i.e.*, $f(x) = \mathbf{w}^\top x$. In summary, the AUC maximization can be formulated by

$$\begin{aligned} & \mathrm{argmin}_{\|\mathbf{w}\| \leq R} \, \mathbb{E}\Big[(1 - \mathbf{w}^\top(x - x'))^2|y = 1, y' = -1\Big] \\ = \; & \mathrm{argmin}_{\|\mathbf{w}\| \leq R} \, \tfrac{1}{p(1-p)} \iint_{\mathcal{Z} \times \mathcal{Z}} (1 - \mathbf{w}^\top(x - x'))^2 \mathbb{I}_{[y=1]}\mathbb{I}_{[y'=-1]} d\rho(z)d\rho(z'). \end{aligned} \tag{3}$$

When $\rho$ is a uniform distribution over training data $\mathbf{z}$, we obtain the empirical minimization (ERM) problem for AUC optimization studied in [6, 21][1]

$$\underset{\|\mathbf{w}\| \leq R}{\operatorname{argmin}} \frac{1}{n^+ n^-} \sum_{i=1}^{n} \sum_{j=1}^{n} (1 - \mathbf{w}^\top (x_i - x_j))^2 \mathbb{I}_{[y_i=1 \wedge y_j=-1]}, \tag{4}$$

where $n^+$ and $n^-$ denote the numbers of instances in the positive and negative classes, respectively.

## 2.1 Equivalent Representation as a (Stochastic) Saddle Point Problem (SPP)

The main result of this work is the equivalence of problem (3) to a stochastic Saddle Point Problem (SPP) (*e.g.*, [14]). A stochastic SPP is generally in the form of

$$\min_{u \in \Omega_1} \max_{\alpha \in \Omega_2} \big\{ f(u, \alpha) := \mathbb{E}[F(u, \alpha, \xi)] \big\}, \tag{5}$$

where $\Omega_1 \subseteq \mathbb{R}^d$ and $\Omega_2 \subseteq \mathbb{R}^m$ are nonempty closed convex sets, $\xi$ is a random vector with non-empty measurable set $\Xi \subseteq \mathbb{R}^p$, and $F: \Omega_1 \times \Omega_2 \times \Xi \to \mathbb{R}$. Here $\mathbb{E}[F(u, \alpha, \xi)] = \int_{\Xi} F(u, \alpha, \xi) d \Pr(\xi)$, and function $f(u, \alpha)$ is convex in $u \in \Omega_1$ and concave in $\alpha \in \Omega_2$. In general, $u$ and $\alpha$ are referred to as the primal variable and the dual variable, respectively.

The following theorem shows that (3) is equivalent to a stochastic SPP (5). First, define $F: \mathbb{R}^d \times \mathbb{R}^3 \times \mathcal{Z} \to \mathbb{R}$, for any $\mathbf{w} \in \mathbb{R}^d$, $a, b, \alpha \in \mathbb{R}$ and $z = (x, y) \in \mathcal{Z}$, by

$$F(\mathbf{w}, a, b, \alpha; z) = (1-p)(\mathbf{w}^\top x - a)^2 \mathbb{I}_{[y=1]} + p(\mathbf{w}^\top x - b)^2 \mathbb{I}_{[y=-1]}$$
$$+ 2(1+\alpha)(p\mathbf{w}^\top x \mathbb{I}_{[y=-1]} - (1-p)\mathbf{w}^\top x \mathbb{I}_{[y=1]}) - p(1-p)\alpha^2. \tag{6}$$

**Theorem 1.** *The AUC optimization* (3) *is equivalent to*

$$\min_{\substack{\|\mathbf{w}\| \leq R \\ (a,b) \in \mathbb{R}^2}} \max_{\alpha \in \mathbb{R}} \Big\{ f(\mathbf{w}, a, b, \alpha) := \int_{\mathcal{Z}} F(\mathbf{w}, a, b, \alpha; z) d\rho(z) \Big\}. \tag{7}$$

*Proof.* It suffices to prove the claim that the objective function of (3) equals to $1 + \min_{(a,b) \in \mathbb{R}^2} \max_{\alpha \in \mathbb{R}} \int_{\mathcal{Z}} F(\mathbf{w}, a, b, \alpha; z) d\rho(z)$.

To this end, note that $z = (x, y)$ and $z = (x', y')$ are samples independently drawn from $\rho$. Therefore, the objective function of (3) can be rewritten as

$$\mathbb{E}\big[(1 - \mathbf{w}^\top (x - x'))^2 | y = 1, y' = -1\big] = 1 + \mathbb{E}[(\mathbf{w}^\top x)^2 | y = 1] + \mathbb{E}[(\mathbf{w}^\top x')^2 | y' = -1]$$
$$- 2\mathbb{E}[\mathbf{w}^\top x | y = 1] + 2\mathbb{E}[\mathbf{w}^\top x' | y' = -1] - 2(\mathbb{E}[\mathbf{w}^\top x | y = 1])(\mathbb{E}[\mathbf{w}^\top x' | y' = -1])$$
$$= 1 + \big\{ \mathbb{E}[(\mathbf{w}^\top x)^2 | y = 1] - (\mathbb{E}[\mathbf{w}^\top x | y = 1])^2 \big\} + \big\{ \mathbb{E}[(\mathbf{w}^\top x')^2 | y' = -1] - (\mathbb{E}[\mathbf{w}^\top x' | y' = -1])^2 \big\}$$
$$- 2\mathbb{E}[\mathbf{w}^\top x | y = 1] + 2\mathbb{E}[\mathbf{w}^\top x' | y' = -1] + (\mathbb{E}[\mathbf{w}^\top x | y = 1] - \mathbb{E}[\mathbf{w}^\top x' | y' = -1])^2. \tag{8}$$

Note that $\mathbb{E}[(\mathbf{w}^\top x)^2 | y = 1] - (\mathbb{E}[\mathbf{w}^\top x | y = 1])^2 = \frac{1}{p} \int_{\mathcal{Z}} (\mathbf{w}^\top x)^2 \mathbb{I}_{[y=1]} d\rho(z) - (\frac{1}{p} \int_{\mathcal{Z}} \mathbf{w}^\top x \mathbb{I}_{[y=1]} d\rho(z))^2 = \min_{a \in \mathbb{R}} \frac{1}{p} \int_{\mathcal{Z}} (\mathbf{w}^\top x - a)^2 \mathbb{I}_{[y=1]} d\rho(z) = \min_{a \in \mathbb{R}} \mathbb{E}[(\mathbf{w}^\top x - a)^2 | y = 1]$, where the minimization is achieved by

$$a = \mathbb{E}[\mathbf{w}^\top x | y = 1]. \tag{9}$$

Likewise, $\min_b \mathbb{E}[(\mathbf{w}^\top x' - b)^2 | y' = -1] = \mathbb{E}[(\mathbf{w}^\top x')^2 | y' = -1] - (\mathbb{E}[\mathbf{w}^\top x' | y' = -1])^2$ where the minimization is obtained by letting

$$b = \mathbb{E}[\mathbf{w}^\top x' | y' = -1]. \tag{10}$$

Moreover, observe that $(\mathbb{E}[\mathbf{w}^\top x | y = 1] - \mathbb{E}[\mathbf{w}^\top x' | y' = -1])^2 = \max_\alpha \big\{ 2\alpha(\mathbb{E}[\mathbf{w}^\top x' | y' = -1] - \mathbb{E}[\mathbf{w}^\top x | y = 1]) - \alpha^2 \big\}$, where the maximization is achieved with

$$\alpha = \mathbb{E}[\mathbf{w}^\top x' | y' = -1] - \mathbb{E}[\mathbf{w}^\top x | y = 1]. \tag{11}$$

Putting all these equalities into (8) implies that

$$\mathbb{E}\left[(1 - \mathbf{w}^\top(x - x'))^2 | y = 1, y' = -1\right] = 1 + \min_{(a,b)\in\mathbb{R}^2} \max_{\alpha\in\mathbb{R}} \frac{\int_{\mathcal{Z}} F(\mathbf{w}, a, b; z) d\rho(z)}{p(1 - p)}.$$

This proves the claim and hence the theorem. □

In addition, we can prove the following result.

**Proposition 1.** *Function* $f(\mathbf{w}, a, b, \alpha)$ *is convex in* $(\mathbf{w}, a, b) \in \mathbb{R}^{d+2}$ *and concave in* $\alpha \in \mathbb{R}$.

The proof of this proposition can be found in the Supplementary Materials.

## 2.2 Stochastic Online Algorithm for AUC Maximization

The optimal solution to an SPP problem is called a saddle point. Stochastic first-order methods are widely used to get such an optimal saddle point. The main idea of such algorithms (e.g. [13, 14] is to use an *unbiased stochastic estimator* of the true gradient to perform, at each iteration, gradient descent in the primal variable and gradient ascent in the dual variable.

Using the stochastic SPP formulation (7) for AUC optimization, we can develop stochastic on-line learning algorithms which only need to pass the data once. For notational simplicity, let vector $v = (\mathbf{w}^\top, a, b)^\top \in \mathbb{R}^{d+2}$, and for any $w \in \mathbb{R}^d$, $a, b, \alpha \in \mathbb{R}$ and $z = (x, y) \in \mathcal{Z}$, we denote $f(\mathbf{w}, a, b, \alpha)$ as $f(v, \alpha)$, and $F(\mathbf{w}, a, b, \alpha, z)$ as $F(v, \alpha, z)$. The gradient of the objective function in the stochastic SPP problem (7) is given by a $(d + 3)$-dimensional column vector $g(v, \alpha) = (\partial_v f(v, \alpha), -\partial_\alpha f(v, \alpha))$ and its unbiased stochastic estimator is given, for any $z \in \mathcal{Z}$, by $G(v, \alpha, z) = (\partial_u F(v, \alpha, z), -\partial_\alpha F(v, \alpha, z))$. One could directly deploy the stochastic first-order method in [14] to the stochastic SPP formulation (7) for AUC optimization. However, from the definition of $F$ in (6), this would require the knowledge of the unknown probability $p = \mathrm{Pr}(y = 1)$ *a priori*. To overcome this problem, for any $v^\top = (\mathbf{w}^\top, a, b) \in \mathbb{R}^{d+2}$, $\alpha \in \mathbb{R}$ and $z \in \mathcal{Z}$, let

$$\hat{F}_t(v, \alpha, z) = (1 - \hat{p}_t)(\mathbf{w}^\top x - a)^2 \mathbb{I}_{[y=1]} + \hat{p}_t(\mathbf{w}^\top x - b)^2 \mathbb{I}_{[y=-1]}$$
$$+ 2(1 + \alpha)(\hat{p}_t \mathbf{w}^\top x \mathbb{I}_{[y=-1]} - (1 - \hat{p}_t)\mathbf{w}^\top x \mathbb{I}_{[y=1]}) - \hat{p}_t(1 - \hat{p}_t)\alpha^2. \quad (12)$$

where $\hat{p}_t = \frac{\sum_{i=1}^t \mathbb{I}_{[y_i=1]}}{t}$ at iteration $t$. We propose, at iteration $t$, to use the stochastic estimator

$$\hat{G}_t(v, \alpha, z) = (\partial_v \hat{F}_t(v, \alpha, z), -\partial_\alpha \hat{F}_t(v, \alpha, z)) \quad (13)$$

to replace the unbiased, but practically inaccessible, stochastic estimator $G(v, \alpha, z)$. Assume $\kappa = \sup_{x\in\mathcal{X}} \|x\| < \infty$, and recall that $\|\mathbf{w}\| \leq R$. For any optimal solution $(\mathbf{w}^*, a^*, b^*)$ of the stochastic SPP (7) for AUC optimization, by (9), (10) and (11) we know that $|a^*| = \frac{1}{p}|\int_{\mathcal{Z}}\langle w^*, x\rangle \mathbb{I}_{[y=1]} d\rho(z)| \leq R\kappa$, $|b^*| = \frac{1}{1-p}|\int_{\mathcal{Z}}\langle w^*, x'\rangle \mathbb{I}_{[y'=-1]} d\rho(z')| \leq R\kappa$, and $|\alpha^*| = \left|\frac{1}{1-p}\int_{\mathcal{Z}}\langle w^*, x'\rangle \mathbb{I}_{[y'=-1]} d\rho(z') - \frac{1}{p}\int_{\mathcal{Z}}\langle w^*, x\rangle \mathbb{I}_{[y=1]} d\rho(z)\right| \leq 2R\kappa$. Therefore, we can restrict $(w, a, b)$ and $\alpha$ to the following bounded domains:

$$\Omega_1 = \left\{(\mathbf{w}, a, b) \in \mathbb{R}^{d+2} : \|\mathbf{w}\| \leq R, |a| \leq R\kappa, |b| \leq R\kappa\right\}, \quad \Omega_2 = \left\{\alpha \in \mathbb{R} : |\alpha| \leq 2R\kappa\right\}. \quad (14)$$

In this case, the projection steps (e.g. steps 4 and 5) in Table 1 can be easily computed. The pseudo-code of the online AUC optimization algorithm is described in Table 1, to which we refer as *SOLAM*.

## 3 Analysis

We now present the convergence results of the proposed algorithm for AUC optimization. Let $u = (v, \alpha) = (\mathbf{w}, a, b, \alpha)$. The quality of an approximation solution $(\bar{v}_t, \bar{\alpha}_t)$ to the SPP problem (5) at iteration $t$ is measured by the duality gap:

$$\varepsilon_f(\bar{v}_t, \bar{\alpha}_t) = \max_{\alpha\in\Omega_2} f(\bar{v}_t, \alpha) - \min_{v\in\Omega_1} f(v, \bar{\alpha}_t). \quad (15)$$

| Stochastic Online AUC Maximization (SOLAM) |
| --- |
| 1. Choose step sizes $\{\gamma_t > 0 : t \in \mathbb{N}\}$ |
| 2. Initialize $t = 1$, $v_1 \in \Omega_1$, $\alpha_1 \in \Omega_2$ and let $\hat{p}_0 = 0$, $\bar{v}_0 = 0$, $\bar{\alpha}_0 = 0$ and $\bar{\gamma}_0 = 0$. |
| 3. Receive a sample $z_t = (x_t, y_t)$ and compute $\hat{p}_t = \frac{(t-1)\hat{p}_{t-1} + \mathbb{I}_{[y_t=1]}}{t}$ |
| 4. Update $v_{t+1} = P_{\Omega_1}(v_t - \gamma_t \partial_v \hat{F}_t(v_t, \alpha_t, z_t))$ |
| 5. Update $\alpha_{t+1} = P_{\Omega_2}(\alpha_t + \gamma_t \partial_\alpha \hat{F}_t(v_t, \alpha_t, z_t))$ |
| 6. Update $\bar{\gamma}_t = \bar{\gamma}_{t-1} + \gamma_t$ |
| 7. Update $\bar{v}_t = \frac{1}{\bar{\gamma}_t}(\bar{\gamma}_{t-1}\bar{v}_{t-1} + \gamma_t v_t)$, and $\bar{\alpha}_t = \frac{1}{\bar{\gamma}_t}(\bar{\gamma}_{t-1}\bar{\alpha}_{t-1} + \gamma_t \alpha_t)$ |
| 8. Set $t \leftarrow t + 1$ |

Table 1: Pseudo code of the proposed algorithm. In steps 4 and 5, $P_{\Omega_1}(\cdot)$ and $P_{\Omega_2}(\cdot)$ denote the projection to the convex sets $\Omega_1$ and $\Omega_2$, respectively.

**Theorem 2.** *Assume that samples $\{(x_1, y_1), (x_2, y_2), \ldots, (x_T, y_T)\}$ are i.i.d. drawn from a distribution $\rho$ over $\mathcal{X} \times \mathcal{Y}$, let $\Omega_1$ and $\Omega_2$ be given by (14) and the step sizes given by $\{\gamma_t > 0 : t \in \mathbb{N}\}$. For sequence $\{(\bar{v}_t, \bar{\alpha}_t) : t \in [1, T]\}$ generated by SOLAM (Table (1)), and any $0 < \delta < 1$, with probability $1 - \delta$, the following holds*

$$\varepsilon_f(\bar{v}_T, \bar{\alpha}_T) \le C_\kappa \max(R^2, 1) \sqrt{\ln \frac{4T}{\delta}} \left(\sum_{j=1}^T \gamma_j\right)^{-1} \left[1 + \sum_{j=1}^T \gamma_j^2 + \left(\sum_{j=1}^T \gamma_j^2\right)^{\frac{1}{2}} + \sum_{j=1}^T \frac{\gamma_j}{\sqrt{j}}\right],$$

*where $C_\kappa$ is an absolute constant independent of $R$ and $T$ (see its explicit expression in the proof).*

Denote $f^*$ as the optimum of (7) which, by Theorem 1, is identical to the optimal value of AUC optimization (3). From Theorem 2, the following convergence rate is straightforward.

**Corollary 1.** *Under the same assumptions as in Theorem 2, and $\{\gamma_j = \zeta j^{-\frac{1}{2}} : j \in \mathbb{N}\}$ with constant $\zeta > 0$, with probability $1 - \delta$, it holds $|f(\bar{v}_T, \bar{\alpha}_T) - f^*| \le \varepsilon_f(\bar{u}_T) = \mathcal{O}\left(\frac{\ln T \sqrt{\ln\left(\frac{4T}{\delta}\right)}}{\sqrt{T}}\right).$*

While the above convergence rate is obtained by choosing decaying step sizes, one can establish a similar result when a constant step size is appropriately chosen.

The proof of Theorem 2 requires several lemmas. The first is a standard result from convex online learning [16, 22]. We include its proof in the Supplementary Materials for completeness.

**Lemma 1.** *For any $T \in \mathbb{N}$, let $\{\xi_j : j \in [1, T]\}$ be a sequence of vectors in $\mathbb{R}^m$, and $\tilde{u}_1 \in \Omega$ where $\Omega$ is a convex set. For any $t \in [1, T]$ define $\tilde{u}_{t+1} = P_\Omega(\tilde{u}_t - \xi_t)$. Then, for any $u \in \Omega$, there holds $\sum_{t=1}^T (\tilde{u}_t - u)^\top \xi_t \le \frac{\|\tilde{u}_1 - u\|^2}{2} + \frac{1}{2}\sum_{t=1}^T \|\xi_t\|^2$.*

The second lemma is the Pinelis-Bernstein inequality for martingale difference sequence in a Hilbert space, which is from [15, Theorem 3.4]

**Lemma 2.** *Let $\{S_k : k \in \mathbb{N}\}$ be a martingale difference sequence in a Hilbert space. Suppose that almost surely $\|S_k\| \le B$ and $\sum_{k=1}^T \mathbb{E}[\|S_k\|^2 | S_1, \ldots, S_{k-1}] \le \sigma_T^2$. Then, for any $0 < \delta < 1$, there holds, with probability at least $1 - \delta$, $\sup_{1 \le j \le T} \left\|\sum_{k=1}^j S_k\right\| \le 2\left(\frac{B}{3} + \sigma_T\right)\log\frac{2}{\delta}$.*

The third lemma indicates that the approximate stochastic estimator $\hat{G}_j(u, z)$ defined by (13), is not far away from the unbiased one $G(u, z)$. Its proof is given in the Supplementary materials.

**Lemma 3.** *Let $\Omega_1$ and $\Omega_2$ be given by (14) and denote by $\Omega = \Omega_1 \times \Omega_2$. For any $t \in \mathbb{N}$, with probability $1 - \delta$, there holds $\sup_{u \in \Omega, z \in \mathcal{Z}} \|\hat{G}_t(u, z) - G(u, z)\| \le 2\kappa(4\kappa R + 11R + 1)\left(\ln\left(\frac{2}{\delta}\right)/t\right)^{\frac{1}{2}}.$*

**Proof of Theorem 2.** By the convexity of $f(\cdot, \alpha)$ and concavity of of $f(v, \cdot)$, for any $u = (v, \alpha) \in \Omega_1 \times \Omega_2$, we get $f(v_t, \alpha) - f(v, \alpha_t) = (f(v_t, \alpha_t) - f(v, \alpha_t)) + (f(v_t, \alpha) - f(v_t, \alpha_t)) \le (v_t - v)^\top \partial_v f(v_t, \alpha_t) - (\alpha_t - \alpha)\partial_\alpha f(v_t, \alpha_t) = (u_t - u)^\top g(u_t)$. Hence, there holds

$$\max_{\alpha \in \Omega_2} f(\bar{v}_T, \alpha) - \min_{v \in \Omega_1} f(v, \bar{\alpha}_T) \le \left(\sum_{t=1}^T \gamma_t\right)^{-1}\left(\max_{\alpha \in \Omega_2}\sum_{t=1}^T \gamma_t f(v_t, \alpha) - \min_{v \in \Omega_1}\sum_{t=1}^T \gamma_t f(v, \alpha_t)\right)$$

$$\leq (\sum_{t=1}^{T} \gamma_t)^{-1} \max_{u \in \Omega_1 \times \Omega_2} \sum_{t=1}^{T} \gamma_t (u_t - u)^\top g(u_t) \qquad (16)$$

Recall that $\Omega = \Omega_1 \times \Omega_2$. The steps 4 and 5 in Algorithm SOLAM can be rewritten as $u_{t+1} = (v_{t+1}, \alpha_{t+1}) = P_\Omega(u_t - \gamma_t \hat{G}_t(u_t, z_t))$. By applying Lemma 1 with $\xi_t = \gamma_t \hat{G}_t(u_t, z_t)$, we have, for any $u \in \Omega$, that $\sum_{t=1}^{T} \gamma_t (u_t - u)^\top \hat{G}_t(u_t, z_t) \leq \frac{\|u_1 - u\|^2}{2} + \frac{1}{2} \sum_{t=1}^{T} \gamma_t^2 \|\hat{G}_t(u_t, z_t)\|^2$, which yields that

$$\sup_{u \in \Omega} \sum_{t=1}^{T} \gamma_t (u_t - u)^\top g(u_t) \leq \sup_{u \in \Omega} \frac{\|u_1 - u\|^2}{2} + \frac{1}{2} \sum_{t=1}^{T} \gamma_t^2 \|\hat{G}_t(u_t, z_t)\|^2$$

$$+ \sup_{u \in \Omega} \sum_{t=1}^{T} \gamma_t (u_t - u)^\top (g(u_t) - \hat{G}_t(u_t, z_t)) \leq \sup_{u \in \Omega} \frac{\|u_1 - u\|^2}{2} + \frac{1}{2} \sum_{t=1}^{T} \gamma_t^2 \|\hat{G}_t(u_t, z_t)\|^2$$

$$+ \sup_{u \in \Omega} \sum_{t=1}^{T} \gamma_t (u_t - u)^\top (g(u_t) - G(u_t, z_t)) + \sup_{u \in \Omega} \sum_{t=1}^{T} \gamma_t (u_t - u)^\top (G(u_t, z_t) - \hat{G}_t(u_t, z_t)) \quad (17)$$

Now we estimate the terms on the right hand side of (17) as follows.

For the first term, we have

$$\frac{1}{2} \sup_{u \in \Omega} \|u_1 - u\|^2 \leq 2 \sup_{v \in \Omega_1, \alpha \in \Omega_2} (\|v\|^2 + |\alpha|^2) \leq 2 \sup_{u \in \Omega} \|u\|^2 \leq 2R^2(1 + 6\kappa^2). \qquad (18)$$

For the second term on the right hand side of (17), observe that $\sup_{x \in \mathcal{X}} \|x\| \leq \kappa$ and $u_t = (\mathbf{w}_t, a_t, b_t, \alpha_t) \in \Omega = \{(\mathbf{w}, a, b, \alpha) : \|\mathbf{w}\| \leq R, |a| \leq \kappa R, |b| \leq \kappa R, |\alpha| \leq 2\kappa R\}$. Combining this with the definition of $\hat{G}_t(u_t, z_t)$ given by (13), one can easily get $\|\hat{G}_t(u_t, z_t)\| \leq \|\partial_\mathbf{w} \hat{F}_t(u_t, z_t)\| + |\partial_a \hat{F}_t(u_t, z_t)| + |\partial_b \hat{F}_t(u_t, z_t)| + |\partial_\alpha \hat{F}_t(u_t, z_t)| \leq 2\kappa(2R + 1 + 2R\kappa)$. Hence, there holds

$$\frac{1}{2} \sum_{t=1}^{T} \gamma_t^2 \|\hat{G}_t(u_t, z_t)\|^2 \leq 2\kappa^2 (2R + 1 + 2R\kappa)^2 (\sum_{t=1}^{T} \gamma_t^2). \qquad (19)$$

The third term on the right hand side of (17) can be bounded by $\sup_{u \in \Omega} \sum_{t=1}^{T} \gamma_t (u_t - u)^\top (g(u_t) - G(u_t, z_t)) \leq \sup_{u \in \Omega} [\sum_{t=1}^{T} \gamma_t (\tilde{u}_t - u)^\top (g(u_t) - G(u_t, z_t))] + \sum_{t=1}^{T} \gamma_t (u_t - \tilde{u}_t)^\top (g(u_t) - G(u_t, z_t))$, where $\tilde{u}_1 = 0 \in \Omega$ and $\tilde{u}_{t+1} = P_\Omega(\tilde{u}_t - \gamma_t(g(u_t) - G(u_t, z_t)))$ for any $t \in [1, T]$. Applying Lemma 1 with $\xi_t = \gamma_t(g(u_t) - G(u_t, z_t))$ yields that

$$\sup_{u \in \Omega} \sum_{t=1}^{T} \gamma_t (\tilde{u}_t - u)^\top (g(u_t) - G(u_t, z_t)) \leq \sup_{u \in \Omega} \frac{\|u\|^2}{2} + \frac{1}{2} \sum_{t=1}^{T} \gamma_t^2 \|g(u_t) - G(u_t, z_t)\|^2$$

$$\leq \frac{1}{2} R^2(1 + 6\kappa^2) + 4\kappa^2(2R + 1 + 2R\kappa)^2 \sum_{t=1}^{T} \gamma_t^2, \quad (20)$$

where we used $\|G(u_t, z_t)\|$ and $\|g(u_t)\|$ is uniformly bounded by $2\kappa(2R + 1 + 2R\kappa)$. Notice that $u_t$ and $\tilde{u}_t$ are only dependent on $\{z_1, z_2, \ldots, z_{t-1}\}$, $\{S_t = \gamma_t(u_t - \tilde{u}_t)^\top (g(u_t) - G(u_t, z_t)) : t = 1, \ldots, t\}$ is a martingale difference sequence. Observe that $\mathbb{E}[\|S_t\|^2 | z_1, \ldots, z_{t-1}] = \gamma_t^2 \iint_{\mathcal{Z}} ((u_t - \tilde{u}_t)^\top (g(u_t) - G(u_t, z)))^2 d\rho(z) \leq \gamma_t^2 \sup_{u \in \Omega, z \in \mathcal{Z}} [\|u_t - \tilde{u}_t\|^2 \|g(u_t) - G(u_t, z_t)\|^2] \leq \gamma_t^2 [2\kappa R \sqrt{1 + 6\kappa^2}(2R + 1 + 2R\kappa)]^2$. Applying Lemma 2 with $\sigma_T^2 = [2\kappa R \sqrt{1 + 6\kappa^2}(2R + 1 + 2R\kappa)]^2 \sum_{t=1}^{T} \gamma_t^2$, $B = \sup_{t=1}^{T} \gamma_t |(u_t - \tilde{u}_t)^\top (g(u_t) - G(u_t, z_t))| \leq \sigma_T$ implies that, with probability $1 - \frac{\delta}{2}$, there holds

$$\sum_{t=1}^{T} \gamma_t (u_t - \tilde{u}_t)^\top (g(u_t) - G(u_t, z_t)) \leq \frac{16\kappa R \sqrt{1 + 6\kappa^2}(2R + 1 + 2R\kappa)}{3} \sqrt{\sum_{t=1}^{T} \gamma_t^2}. \qquad (21)$$

Combining (20) with (21) implies, with probability $1 - \frac{\delta}{2}$,

$$\sup_{u \in \Omega} \sum_{t=1}^{T} \gamma_t (u_t - u)^\top (g(u_t) - G(u_t, z_t)) \leq \frac{R^2(1 + 6\kappa^2)}{2} + 4\kappa^2(2R + 1 + 2R\kappa)^2 \sum_{t=1}^{T} \gamma_t^2$$

| datasets | ♯inst | ♯feat | datasets | ♯inst | ♯feat | datasets | ♯inst | ♯feat | datasets | ♯inst | ♯feat |
|---|---|---|---|---|---|---|---|---|---|---|---|
| diabetes | 768 | 8 | fourclass | 862 | 2 | german | 1,000 | 24 | splice | 3,175 | 60 |
| usps | 9,298 | 256 | a9a | 32,561 | 123 | mnist | 60,000 | 780 | acoustic | 78,823 | 50 |
| ijcnn1 | 141,691 | 22 | covtype | 581,012 | 54 | sector | 9,619 | 55,197 | news20 | 15,935 | 62,061 |

Table 2: *Basic information about the benchmark datasets used in the experiments.*

$$+ \frac{16\kappa R\sqrt{1+6\kappa^2}(2R+1+2R\kappa)}{3}\Big(\sum_{t=1}^{T}\gamma_t^2\Big)^{1/2}. \quad (22)$$

By Lemma 3, for any $t \in [1,T]$ there holds, with probability $1-\frac{\delta}{2T}$, $\sup_{u\in\Omega, z\in\mathcal{Z}}\|\hat{G}_t(u,z)-G(u,z)\| \leq$

$2\kappa(2R(\kappa+1)+1)\sqrt{\ln\big(\frac{4T}{\delta}\big)/t}$. Hence, the fourth term on the righthand side of (17) can estimated as follows: with probability $1-\frac{\delta}{2}$, there holds

$$\sup_{u\in\Omega}\sum_{t=1}^{T}\gamma_t(u_t-u)^\top(G(u_t,z_t)-\hat{G}_t(u_t,z_t)) \leq 2\sup_{u\Omega}\|u\|\Big(\sum_{t=1}^{T}\gamma_t\sup_{u\in\Omega,z\in\mathcal{Z}}\|\hat{G}_t(u,z)-G(u,z)\|\Big)$$

$$\leq 8R\kappa(4R\kappa+11R+1)\sqrt{6\kappa^2+1}\sum_{t=1}^{T}\frac{\gamma_t}{\sqrt{t}}. \quad (23)$$

Putting the estimations (18), (19), (22), (23) and (17) back into (16) implies that

$$\varepsilon_f(\bar{u}_T) \leq C_\kappa \max(R^2,1)\sqrt{\ln\frac{4T}{\delta}}\Big(\sum_{t=1}^{T}\gamma_t\Big)^{-1}\Big[1+\sum_{t=1}^{T}\gamma_t^2+\Big(\sum_{t=1}^{T}\gamma_t^2\Big)^{\frac{1}{2}}+\sum_{t=1}^{T}\frac{\gamma_t}{\sqrt{t}}\Big],$$

where $C_\kappa = \frac{5}{2}(1+6\kappa^2)+6\kappa^2(\kappa+3)^2+\frac{112}{3}\kappa\sqrt{6\kappa^2+1}(2\kappa+3)$. $\qquad\square$

# 4   Experiments

In this section, we report experimental evaluations of the SOLAM algorithm and comparing its performance with existing state-of-the-art learning algorithms for AUC optimization. SOLAM was implemented in MATLAB, and MATLAB code of the compared methods were obtained from the authors of corresponding papers. In the training phase, we use five-fold cross validation to determine the initial learning rate $\zeta \in [1:9:100]$ and the bound on $\mathbf{w}$, $R \in 10^{[-1:1:5]}$ by a grid search. Following the evaluation protocol of [6], the performance of SOLAM was evaluated by averaging results from five runs of five-fold cross validations.

Our experiments were performed based on 12 datasets that had been used in previous studies. For multi-class datasets, *e.g.*, news20 and sector, we transform them into binary classification problems by randomly partitioning the data into two groups, where each group includes the same number of classes. Information about these datasets is summarized in Table 2.

On these datasets, we evaluate and compare SOLAM with four online and two offline learning algorithms for AUC maximization, *i.e.* *one-pass AUC maximization* (OPAUC) [6], which uses the $\ell_2$ loss surrogate of the AUC objective function; *online AUC maximization* [21] that uses the hinge loss surrogate of the AUC objective function with two variants, one with sequential update (OAMseq) and the other using gradient update (OAMgra); *online Uni-Exp* [12] which uses the weighted univariate exponential loss; B-SVM-OR [10], which is a batch learning algorithm using the hinge loss surrogate of the AUC objective function; and B-LS-SVM, which is a batch learning algorithm using the $\ell_2$ loss surrogate of the AUC objective function.

Classification performances on the testing dataset of all methods are given in Table 3. These results show that SOLAM achieves similar performances as other state-of-the-art online and offline methods based on AUC maximization. The performance of SOLAM is better than the offline methods on *acoustic* and *covtype* which could be due to the normalization of features used in our experiments for SOLAM. On the other hand, the main advantage of SOLAM is the running efficiency, as we pointed out in the Introduction, its per-iteration running time and space complexity is linear in data dimension and do not depend on the iteration number. In Figure 1, we show AUC vs. run time (seconds) for

| Datasets | SOLAM | OPAUC | OAM$_{seq}$ | OAM$_{gra}$ | online Uni-Exp | B-SVM-OR | B-LS-SVM |
|---|---|---|---|---|---|---|---|
| diabetes | .8253±.0314 | .8309±.0350 | .8264±.0367 | .8262±.0338 | .8215±.0309 | .8326±.0328 | .8325±.0329 |
| fourclass | .8226±.0240 | .8310±.0251 | .8306±.0247 | .8295±.0251 | .8281±.0305 | .8305±.0311 | .8309±.0309 |
| german | .7882±.0243 | .7978±.0347 | .7747±.0411 | .7723±.0358 | .7908±.0367 | .7935±.0348 | .7994±.0343 |
| splice | .9253±.0097 | .9232±.0099 | .8594±.0194 | .8864±.0166 | .8931±.0213 | .9239±.0089 | .9245±.0092 |
| usps | .9766±.0032 | .9620±.0040 | .9310±.0159 | .9348±.0122 | .9538±.0045 | .9630±.0047 | .9634±.0045 |
| a9a | .9001±.0042 | .9002±.0047 | .8420±.0174 | .8571±.0173 | .9005±.0024 | .9009±.0036 | .8982±.0028 |
| mnist | .9324±.0020 | .9242±.0021 | .8615±.0087 | .8643±.0112 | .7932±.0245 | .9340±.0020 | .9336±.0025 |
| acoustic | .8898±.0026 | .8192±.0032 | .7113±.0590 | .7711±.0217 | .8171±.0034 | .8262±.0032 | .8210±.0033 |
| ijcnn1 | .9215±.0045 | .9269±.0021 | .9209±.0079 | .9100±.0092 | .9264±.0035 | .9337±.0024 | .9320±.0037 |
| covtype | .9744±.0004 | .8244±.0014 | .7361±.0317 | .7403±.0289 | .8236±.0017 | .8248±.0013 | .8222±.0014 |
| sector | .9834±.0023 | .9292±.0081 | .9163±.0087 | .9043±.0100 | .9215±.0034 | - | - |
| news20 | .9467±.0039 | .8871±.0083 | .8543±.0099 | .8346±.0094 | .8880±.0047 | - | - |

Table 3: *Comparison of the testing AUC values (mean±std.) on the evaluated datasets. To accelerate the experiments, the performances of OPAUC, OAMseq, OAMgra, online Uni-Exp, B-SVM-OR and B-LS-SVM were taken from [6]*

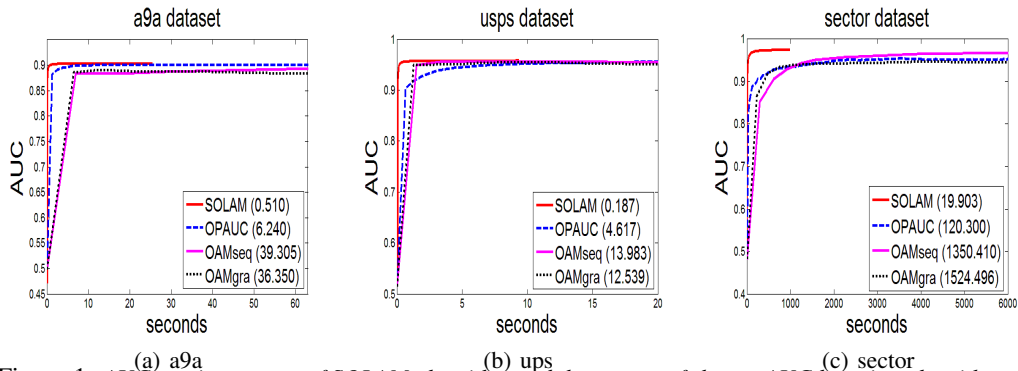

(a) a9a          (b) ups          (c) sector

Figure 1: *AUC vs. time curves of SOLAM algorithm and three state-of-the-art AUC learning algorithms,* i.e., *OPAUC [6], OAMseq [21], and OAMgra [21]. The values in parentheses indicate the average running time (seconds) per pass for each algorithm.*

SOLAM and three other state-of-the-art online learning algorithms,*i.e.*, OPAUC [6], OAMseq [21], and OAMgra [21] over three datasets (a9a, ups, and sector), along with the per-iteration running time in the legend[2]. These results show that SOLAM in general reaches convergence faster in comparison of, while achieving competitive performance.

## 5 Conclusion

In this paper we showed that AUC maximization is equivalent to a stochastic saddle point problem, from which we proposed a novel online learning algorithm for AUC optimization. In contrast to the existing algorithms [6, 21], the main advantage of our algorithm is that it does not need to store all previous examples nor its second-order covariance matrix. Hence, it is a truly online learning algorithm with one-datum space and per-iteration complexities, which are the same as online gradient descent algorithms [22] for classification.

There are several research directions for future work. Firstly, the convergence rate $\mathcal{O}(1/\sqrt{T})$ for SOLAM only matches that of the black-box sub-gradient method. It would be interesting to derive fast convergence rate $\mathcal{O}(1/T)$ by exploring the special structure of the objective function $F$ defined by (6). Secondly, the convergence was established using the duality gap associated with the stochastic SPP formulation 7. It would be interesting to establish the strong convergence of the output $\bar{w}_T$ of algorithm SOLAM to its optimal solution of the actual AUC optimization problem (3). Thirdly, the SPP formulation (1) holds for the least square loss. We do not know if the same formulation holds true for other loss functions such as the logistic regression or the hinge loss.

## Footnotes

[1]The work [6, 21] studied the regularized ERM problem, *i.e.* $\min_{\mathbf{w} \in \mathbb{R}^d} \frac{1}{n^+ n^-} \sum_{i=1}^{n} \sum_{j=1}^{n} (1 - \mathbf{w}^\top (x_i - x_j))^2 \mathbb{I}_{[y_i=1]} \mathbb{I}_{[y_j=-1]} + \frac{\lambda}{2} \|\mathbf{w}\|^2$, which is equivalent to (3) with $\Omega$ being a bounded ball in $\mathbb{R}^d$.

[2]Experiments were performed with running time reported based on a workstation with 12 nodes, each with an Intel Xeon E5-2620 2.0GHz CPU and 64GB RAM.

# References

[1] F. R. Bach and E. Moulines. Non-asymptotic analysis of stochastic approximation algorithms for machine learning. In *NIPS*, 2011.

[2] L. Bottou and Y. LeCun. Large scale online learning. In *NIPS*, 2003.

[3] N. Cesa-Bianchi, A. Conconi, and C. Gentile. On the generalization ability of on-line learning algorithms. *IEEE Trans. Information Theory*, 50(9):2050–2057, 2004.

[4] S. Clemencon, G. Lugosi, and N. Vayatis. Ranking and empirical minimization of u-statistics. *The Annals of Statistics*, 36(2):844–874, 2008.

[5] C. Cortes and M. Mohri. AUC optimization vs. error rate minimization. In *NIPS*, 2003.

[6] W. Gao, R. Jin, S. Zhu, and Z. H. Zhou. One-pass AUC optimization. In *ICML*, 2013.

[7] W. Gao and Z.H. Zhou. On the consistency of AUC pairwise optimization. In *International Joint Conference on Artificial Intelligence*, 2015.

[8] J. A. Hanley and B. J. McNeil. The meaning and use of the area under of receiver operating characteristic (roc) curve. *Radiology*, 143(1):29–36, 1982.

[9] T. Joachims. A support vector method for multivariate performance measures. In *ICML*, 2005.

[10] Thorsten Joachims. Training linear svms in linear time. In *Proceedings of the Twelfth ACM SIGKDD International Conference on Knowledge Discovery and Data Mining*, pages 217–226, 2006.

[11] P. Kar, B. K. Sriperumbudur, P. Jain, and H. Karnick. On the generalization ability of online learning algorithms for pairwise loss functions. In *ICML*, 2013.

[12] W. Kotlowski, K. Dembczynski, and E. Hüllermeier. Bipartite ranking through minimization of univariate loss. In *ICML*, 2011.

[13] G. Lan. An optimal method for stochastic composite optimization. *Math Programming*, 133(1-2):365–397, 2012.

[14] A. Nemirovski, A. Juditsky, G. Lan, and A. Shapiro. Robust stochastic approximation approach to stochastic programming. *SIAM Journal on Optimization*, 19(4):1574–1609, 2009.

[15] I. Pinelis. Optimum bounds for the distributions of martingales in banach spaces. *The Annals of Probability*, 22(4):1679–1706, 1994.

[16] A. Rakhlin, O. Shamir, and K. Sridharan. Making gradient descent optimal for strongly convex stochastic optimization. In *ICML*, 2012.

[17] A. Rakotomamonjy. Optimizing area under roc curve with svms. In *1st International Workshop on ROC Analysis in Artificial Intelligence*, 2004.

[18] Y. Wang, R. Khardon, D. Pechyony, and R. Jones. Generalization bounds for online learning algorithms with pairwise loss functions. In *COLT*, 2012.

[19] Y. Ying and M. Pontil. Online gradient descent learning algorithms. *Foundations of Computational Mathematics*, 8(5):561–596, 2008.

[20] Y. Ying and D. X. Zhou. Online pairwise learning algorithms. *Neural Computation*, 28:743–777, 2016.

[21] P. Zhao, S. C. H. Hoi, R. Jin, and T. Yang. Online AUC maximization. In *ICML*, 2011.

[22] M. Zinkevich. Online convex programming and generalized infinitesimal gradient ascent. In *ICML*, 2003.

